# Switching state space model for simultaneously estimating state transitions and nonstationary firing rates

## Abstract

We propose an algorithm for simultaneously estimating state transitions among neural states, the number of neural states, and nonstationary firing rates using a switching state space model (SSSM). This algorithm enables us to detect state transitions on the basis of not only the discontinuous changes of mean firing rates but also discontinuous changes in temporal profiles of firing rates, e.g., temporal correlation. We construct a variational Bayes algorithm for a non-Gaussian SSSM whose non-Gaussian property is caused by binary spike events. Synthetic data analysis reveals that our algorithm has the high performance for estimating state transitions, the number of neural states, and nonstationary firing rates compared to previous methods. We also analyze neural data that were recorded from the medial temporal area. The statistically detected neural states probably coincide with transient and sustained states that have been detected heuristically. Estimated parameters suggest that our algorithm detects the state transition on the basis of discontinuous changes in the temporal correlation of firing rates, which transitions previous methods cannot detect. This result suggests that our algorithm is advantageous in real-data analysis.

## 1 Introduction

Elucidating neural encoding is one of the most important issues in neuroscience. Recent studies have suggested that cortical neuron activities transit among neural states in response to applied sensory stimuli[1-3]. Abeles *et al*. detected state transitions among neural states using a hidden Markov model whose output distribution is multivariate Poisson distribution (multivariate-Poisson hidden Markov model(mPHMM))[1]. Kemere *et al*. indicated the correspondence relationship between the time of the state transitions and the time when input properties change[2]. They also suggested that the number of neural states corresponds to the number of input properties. Assessing neural states and their transitions thus play a significant role in elucidating neural encoding. Firing rates have state-dependent properties because mean and temporal correlations are significantly different among all neural states[1]. We call the times of state transitions as change points. Change points are those times when the time-series data statistics change significantly and cause nonstationarity in time-series data. In this study, stationarity means that time-series data have temporally uniform statistical properties. By this definition, data that do not have stationarity have nonstationarity.

Previous studies have detected change points on the basis of discontinuous changes in mean firing rates using an mPHMM. In this model, firing rates in each neural state take a constant value. However, actually in motor cortex, average firing rates and preferred direction change dynamically in motor planning and execution[4]. This makes it necessary to estimate state-dependent, instantaneous firing rates. On the other hand, when place cells burst within their place field[5], the inter-burst

intervals correspond to the $\theta$ rhythm frequency. Medial temporal (MT) area neurons show oscillatory firing rates when the target speed is modulated in the manner of a sinusoidal function[6]. These results indicate that change points also need to be detected when the temporal profiles of firing rates change discontinuously.

One solution is to simultaneously estimate both change points and instantaneous firing rates. A switching state space model(SSSM)[7] can model nonstationary time-series data that include change points. An SSSM defines two or more system models, one of which is modeled to generate observation data through an observation model. It can model nonstationary time-series data while switching system models at change points. Each system model estimates stationary state variables in the region that it handles. Recent studies have been focusing on constructing algorithms for estimating firing rates using single-trial data to consider trial-by-trial variations in neural activities [8]. However, these previous methods assume firing rate stationarity within a trial. They cannot estimate nonstationary firing rates that include change points. An SSSM may be used to estimate nonstationary firing rates using single-trial data.

We propose an algorithm for simultaneously estimating state transitions among neural states and nonstationary firing rates using an SSSM. We expect to be able to estimate change points when not only mean firing rates but also temporal profiles of firing rates change discontinuously. Our algorithm consists of a non-Gaussian SSSM, whose non-Gaussian property is caused by binary spike events. Learning and estimation algorithms consist of variational Bayes[9,10] and local variational methods[11,12]. Automatic relevance determination (ARD) induced by the variational Bayes method[13] enables us to estimate the number of neural states after pruning redundant ones. For simplicity, we focus on analyzing single-neuron data. Although many studies have discussed state transitions by analyzing multi-neuron data, some of them have suggested that single-neuron activities reflect state transitions in a recurrent neural network[14]. Note that we can easily extend our algorithm to multi-neuron analysis using the often-used assumption that change points are common among recorded neurons[1-3].

## 2 Definitions of Probabilistic Model

### 2.1 Likelihood Function

Observation time $T$ consists of $K$ time bins of widths $\Delta$ (ms), and each bin includes at most one spike ($\Delta \ll 1$). The spike timings are $\boldsymbol{t} = \{t_1, ..., t_S\}$ where $S$ is the total number of observed spikes. We define $\eta_k$ such that $\eta_k = +1$ if the $k$th bin includes a spike and $\eta_k = -1$ otherwise ($k = 1, ..., K$). The likelihood function is defined by the Bernoulli distribution

$$p(\boldsymbol{t}|\boldsymbol{\lambda}) = \prod_{k=1}^{K}(\lambda_k\Delta)^{\frac{1+\eta_k}{2}}(1-\lambda_k\Delta)^{\frac{1-\eta_k}{2}}, \tag{1}$$

where $\boldsymbol{\lambda} = \{\lambda_1, ..., \lambda_K\}$ and $\lambda_k$ is the firing rate at the $k$th bin. The product of firing rates and bin width corresponds to the spike-occurrence probability and $\lambda_k\Delta \in [0, 1)$ since $\Delta \ll 1$. The logit transformation of $\exp(2x_k) = \frac{\lambda_k\Delta}{1-\lambda_k\Delta}$ ($x_k \in (-\infty, \infty)$) lets us consider the nonnegativity of firing rates in detail[11]. Hereinafter, we call $\boldsymbol{x} = \{x_1, ..., x_K\}$ the "firing rates".

Since $K$ is a large because $\Delta \ll 1$, the computational cost and memory accumulation do matter. We thus use coarse graining[15]. Observation time $T$ consists of $M$ coarse bins of widths $r = C\Delta$ (ms). A coarse bin includes many spikes and the firing rate in each bin is constant. The likelihood function which is obtained by applying the logit transformation and the coarse graining to eq. (1) is

$$p(\boldsymbol{t}|\boldsymbol{x}) = \prod_{m=1}^{M}[\exp(\hat{\eta}_m x_m - C\log 2\cosh x_m)], \tag{2}$$

where $\hat{\eta}_m = \sum_{u=1}^{C} \eta_{(m-1)C+u}$.

### 2.2 Switching State Space Model

An SSSM consists of $N$ system models; for each model, we define a prior distribution. We define label variables $z_m^n$ such that $z_m^n = 1$ if the $n$th system model generates an observation in the $m$th bin and $z_m^n = 0$ otherwise ($n = 1, ..., N, m = 1, ..., M$).

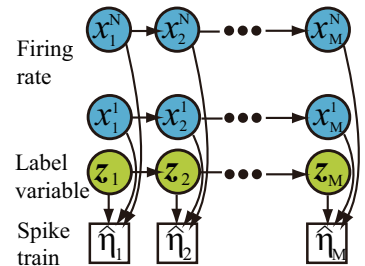

Figure 1: Graphical model representation of SSSM.

We call $N$ the number of labels and the $n$th system model the $n$th label. The joint distribution is defined by

$$p(\boldsymbol{t}, \boldsymbol{x}, \boldsymbol{z}|\boldsymbol{\theta}') = p(\boldsymbol{t}|\boldsymbol{x}, \boldsymbol{z})p(\boldsymbol{z}|\boldsymbol{\pi}, \boldsymbol{a})p(\boldsymbol{x}|\boldsymbol{\mu}, \boldsymbol{\beta}), \tag{3}$$

where $\boldsymbol{x} = \{\boldsymbol{x}^1, ..., \boldsymbol{x}^N\}$, $\boldsymbol{x}^n = \{x_1^n, ..., x_M^n\}$, $\boldsymbol{z} = \{z_1^1, .., z_M^1, ..., z_1^N, ..., z_M^N\}$, and $\boldsymbol{\theta}' = \{\boldsymbol{\pi}, \boldsymbol{a}, \boldsymbol{\mu}, \boldsymbol{\beta}\}$ are parameters. The likelihood function, including label variables, is given by

$$p(\boldsymbol{t}|\boldsymbol{x}, \boldsymbol{z}) = \prod_{n=1}^{N} \prod_{m=1}^{M} [\exp(\hat{\eta}_m x_m^n - C \log 2 \cosh x_m^n)]^{z_m^n}. \tag{4}$$

We define the prior distributions of label variables as

$$p(\boldsymbol{z}_1|\boldsymbol{\pi}) = \prod_{n=1}^{N} (\pi^n)^{z_1^n} \delta(\sum_{n=1}^{N} \pi^n - 1), \tag{5}$$

$$p(\boldsymbol{z}_{m+1}|\boldsymbol{z}_m, \boldsymbol{a}) = \prod_{n=1}^{N} \prod_{k=1}^{N} (a^{nk})^{z_m^n z_{m+1}^k} \delta(\sum_{k=1}^{N} a^{nk} - 1), \tag{6}$$

where $\pi^n$ and $a^{nk}$ are the probabilities that the $n$th label is selected at the initial time and that the $n$th label switches to the $k$th one, respectively. The prior distributions of firing rates are Gaussian

$$p(\boldsymbol{x}) = \prod_{n=1}^{N} p(\boldsymbol{x}^n|\beta^n, \boldsymbol{\mu}^n) = \prod_{n=1}^{N} \sqrt{\frac{|\beta^n \boldsymbol{\Lambda}|}{(2\pi)^M}} \exp(-\frac{\beta^n}{2}(\boldsymbol{x}^n - \boldsymbol{\mu}^n)^T \boldsymbol{\Lambda}(\boldsymbol{x}^n - \boldsymbol{\mu}^n)), \tag{7}$$

where $\beta^n, \boldsymbol{\mu}^n$ respectively mean the temporal correlation and the mean values of the $n$th-label firing rates ($n = 1, ..., N$). Here for simplicity, we introduced $\boldsymbol{\Lambda}$, which is the structure of the temporal correlation satisfying $p(\boldsymbol{x}^n|\beta^n, \boldsymbol{\mu}^n) \propto \prod_m \exp(-\frac{\beta^n}{2}((x_m - \mu_m) - (x_{m-1} - \mu_{m-1}))^2)$. Figure 1 depicts a graphical model representation of an SSSM.

Ghahramani & Hinton (2000) did not introduce *a priori* knowledge about the label switching frequencies. However, in many cases, the time scale of state transitions is probably slower than that of the temporal variation of firing rates. We define prior distributions of $\boldsymbol{\pi}$ and $\boldsymbol{a}$ to introduce *a priori* knowledge about label switching frequencies using Dirichlet distributions

$$p(\boldsymbol{\pi}|\gamma^n) = C(\gamma^n) \prod_{n=1}^{N} (\pi^n)^{\gamma^n - 1} \delta(\sum_{n=1}^{N} \pi^n - 1), \tag{8}$$

$$p(\boldsymbol{a}|\gamma^{nk}) = \prod_{n=1}^{N} \left[ C(\gamma^{nk}) \prod_{k=1}^{N} (a^{nk})^{\gamma^{nk} - 1} \delta(\sum_{k=1}^{N} a^{nk} - 1) \right], \tag{9}$$

where $C(\gamma^n) = \frac{\Gamma(\sum_{n=1}^{N} \gamma^n)}{\Gamma(\gamma^1)...\Gamma(\gamma^N)}$, $C(\gamma^{nk}) = \frac{\Gamma(\sum_{k=1}^{N} \gamma^{nk})}{\Gamma(\gamma^{n1})...\Gamma(\gamma^{nN})}$. $C(\gamma^n)$ and $C(\gamma^{nk})$ correspond to the normalization constants of $p(\boldsymbol{\pi}|\gamma^n)$ and $p(\boldsymbol{a}|\gamma^{nk})$, respectively. $\Gamma(u)$ is the gamma function defined by $\Gamma(u) = \int_0^{\infty} dt t^{u-1} \exp(-t)$. $\gamma^n, \gamma^{nk}$ are hyperparameters to control the probability that the $n$th label is selected at the initial time and that the $n$th label switches to the $k$th. We define the prior distributions of $\boldsymbol{\mu}^n$ and $\beta^n$ using non-informative priors. Since we do not have *a priori* knowledge about neural states, $\boldsymbol{\mu}$ and $\boldsymbol{\beta}$, which characterize each neural state, should be estimated from scratch.

## 3   Estimation and Learning of non-Gaussian SSSM

It is generally computationally difficult to calculate the marginal posterior distribution in an SSSM[6]. We thus use the variational Bayes method to calculate approximated posterior distributions $q(\boldsymbol{w})$ and $q(\boldsymbol{\theta})$ that minimize the variational free energy

$$\mathcal{F}[q] = \int \int d\boldsymbol{w} d\boldsymbol{\theta} q(\boldsymbol{w}) q(\boldsymbol{\theta}) \log \frac{q(\boldsymbol{w})q(\boldsymbol{\theta})}{p(\boldsymbol{t}, \boldsymbol{w}, \boldsymbol{\theta})} = \mathcal{U}[q] - \mathcal{S}[q] \tag{10}$$

where $\boldsymbol{w} = \{\boldsymbol{z}, \boldsymbol{x}\}$ are hidden variables, $\boldsymbol{\theta} = \{\boldsymbol{\pi}, \boldsymbol{a}\}$ are parameters,
$\mathcal{U}[q] = -\int \int d\boldsymbol{w} d\boldsymbol{\theta} q(\boldsymbol{w}) q(\boldsymbol{\theta}) \log p(\boldsymbol{t}, \boldsymbol{w}, \boldsymbol{\theta})$ and $\mathcal{S}[q] = -\int \int d\boldsymbol{w} d\boldsymbol{\theta} q(\boldsymbol{w}) q(\boldsymbol{\theta}) \log(q(\boldsymbol{w})q(\boldsymbol{\theta}))$.
We denote $q(\boldsymbol{w})$ and $q(\boldsymbol{\theta})$ as test distributions. The variational free energy satisfies

$$\log p(\boldsymbol{t}) = -\mathcal{F}[q] + \text{KL}(q(\boldsymbol{w})q(\boldsymbol{\theta}) \| p(\boldsymbol{w}, \boldsymbol{\theta}|\boldsymbol{t})), \tag{11}$$

where $\text{KL}(\text{q}(\boldsymbol{w})\text{q}(\boldsymbol{\theta}) \| \text{p}(\boldsymbol{w}, \boldsymbol{\theta}|\boldsymbol{t}))$ is the Kullback-Leibler divergence between test distributions and a posterior distribution $p(\boldsymbol{w}, \boldsymbol{\theta}|\boldsymbol{t})$ defined by $\text{KL}(\text{q}(\boldsymbol{y}) \| \text{p}(\boldsymbol{y}|\boldsymbol{t})) = \int d\boldsymbol{y} q(\boldsymbol{y}) \log \frac{q(\boldsymbol{y})}{p(\boldsymbol{y}|\boldsymbol{t})}$. Since the marginal likelihood $\log p(\boldsymbol{t})$ takes a constant value, the minimization of variational free energy indirectly minimizes Kullback-Leibler divergence. The variational Bayes method requires conjugacy between the likelihood function (eq. (4)) and the prior distribution (eq. (7)). However, eqs. (4) and (7) are not conjugate to each other because of the binary spike events. The local variational method enables us to construct a variational Bayes algorithm for a non-Gaussian SSSM.

## 3.1 Local Variational Method

The local variational method, which was proposed by Jaakola & Jordan[11], approximately transforms a non-Gaussian distribution into a quadratic-form distribution by introducing variational parameters. Watanabe *et al.* have proven the effectiveness of this method in estimating stationary firing rates[12]. The exponential function in eq. (4) includes $f(x_m^n) = \log 2 \cosh x_m^n$, which is a concave function of $y = (x_m^n)^2$. The concavity can be confirmed by showing the negativity of the second-order derivative of $f(x_m^n)$ with respect to $(x_m^n)^2$ for all $x_m^n$. Considering the tangent line of $f(x_m^n)$ with respect to $(x_m^n)^2$ at $(x_m^n)^2 = (\xi_m^n)^2$, we get a lower bound for eq. (4)

$$p_{\boldsymbol{\xi}}(\boldsymbol{t}|\boldsymbol{x},\boldsymbol{z}) = \prod_{n=1}^{N} \prod_{m=1}^{M} [\exp(\hat{\eta}_m x_m^n - C\tfrac{\tanh \xi_m^n}{2\xi_m^n}((x_m^n)^2 - (\xi_m^n)^2)) - C \log 2 \cosh \xi_m^n)]^{z_m^n}, \quad (12)$$

where $\xi_m^n$ is a variational parameter. Equation (12) satisfies the inequality $p_{\boldsymbol{\xi}}(\boldsymbol{t}|\boldsymbol{x},\boldsymbol{z}) \leq p(\boldsymbol{t}|\boldsymbol{x},\boldsymbol{z})$. We use eq. (12) as the likelihood function instead of eq. (4). The conjugacy between eqs. (12) and (7) enables us to construct the variational Bayes algorithm. Using eq. (12), we find that the variational free energy

$$\mathcal{F}_{\boldsymbol{\xi}}[q] = \int \int d\boldsymbol{w} d\boldsymbol{\theta} q(\boldsymbol{w}) q(\boldsymbol{\theta}) \log \frac{q(\boldsymbol{w})q(\boldsymbol{\theta})}{p_{\boldsymbol{\xi}}(\boldsymbol{t},\boldsymbol{w},\boldsymbol{\theta})} = \mathcal{U}_{\boldsymbol{\xi}}[q] - \mathcal{S}[q] \quad (13)$$

satisfies the inequality $\mathcal{F}_{\boldsymbol{\xi}}[q] \geq \mathcal{F}[q]$, where $\mathcal{U}_{\boldsymbol{\xi}}[q] = -\int \int d\boldsymbol{w} d\boldsymbol{\theta} q_{\boldsymbol{\xi}}(\boldsymbol{w}) q_{\boldsymbol{\xi}}(\boldsymbol{\theta}) \log p_{\boldsymbol{\xi}}(\boldsymbol{s},\boldsymbol{w},\boldsymbol{\theta})$. Since the inequality $\log p(\boldsymbol{t},\boldsymbol{x},\boldsymbol{z}) \geq -\mathcal{F}[q] \geq -\mathcal{F}_{\boldsymbol{\xi}}[q]$ is satisfied, the test distributions that minimize $\mathcal{F}_{\boldsymbol{\xi}}[q]$ can indirectly minimize $\mathcal{F}[q]$ which is analytically intractable. Using the EM algorithm to estimate variational parameters improves the approximation accuracy of $\mathcal{F}_{\boldsymbol{\xi}}[q]$[16].

## 3.2 Variational Bayes Method

We assume the test distributions that satisfy the constraints $q(\boldsymbol{w}) = \prod_{n=1}^{N}(q(\boldsymbol{x}^n|\boldsymbol{\mu}^n,\boldsymbol{\beta}^n))q(\boldsymbol{z})$ and $q(\boldsymbol{\theta}) = q(\boldsymbol{\pi})q(\boldsymbol{a})$, where $\boldsymbol{\mu} = \{\boldsymbol{\mu}^1,...,\boldsymbol{\mu}^N\}, \boldsymbol{\beta} = \{\beta^1,...,\beta^N\}$. Under constraints $\int d\boldsymbol{x} q(\boldsymbol{x}|\boldsymbol{\mu},\boldsymbol{\beta}) = 1$, $\sum_{\boldsymbol{z}} q(\boldsymbol{z}) = 1$, $\int d\boldsymbol{\pi} q(\boldsymbol{\pi}) = 1$ and $\int d\boldsymbol{a} q(\boldsymbol{a}) = 1$, we can obtain the test distributions of hidden variables $\boldsymbol{x}^n, \boldsymbol{z}$ that minimize eq. (13) as follows:

$$q(\boldsymbol{x}^n|\boldsymbol{\mu}^n,\beta^n) = \sqrt{\tfrac{|\boldsymbol{W}^n|}{(2\pi)^M}} \exp(-\tfrac{1}{2}(\boldsymbol{x}^n - \hat{\boldsymbol{\mu}}^n)^T \boldsymbol{W}^n(\boldsymbol{x}^n - \hat{\boldsymbol{\mu}}^n)), \quad (14)$$

$$q(\boldsymbol{z}) \propto \prod_{n=1}^{N} \exp(\hat{\pi}^n)^{z_1^n} \prod_{n=1}^{N} \prod_{m=1}^{M} \exp(\hat{b}_m^n)^{z_m^n} \prod_{m=1}^{M-1} \prod_{n=1}^{N} \prod_{k=1}^{N} \exp(\hat{a}_m^{nk})^{z_m^n z_{m+1}^k}, \quad (15)$$

where $\boldsymbol{W}^n = C\boldsymbol{L}^n + \beta^n\boldsymbol{\Lambda}$, $\hat{\boldsymbol{\mu}}^n = (\boldsymbol{W}^n)^{-1}(\boldsymbol{w}^n + \beta^n\boldsymbol{\Lambda}\boldsymbol{\mu}^n)$, $\hat{\pi}^n = \langle \log \pi^n \rangle$, $\hat{b}_m^n = \hat{\eta}_m \langle x_m^n \rangle - \tfrac{C \tanh \xi_m^n}{2\xi_m^n}(\langle(x_m^n)^2\rangle - (\xi_m^n)^2) - C \log 2 \cosh \xi_m^n$, $\hat{a}^{nk} = \langle \log a^{nk} \rangle$, $\boldsymbol{L}^n$ is the diagonal matrix whose $(m,m)$ component is $\langle z_m^n \rangle \tfrac{\tanh \xi_m^n}{\xi_m^n}$, $\boldsymbol{w}^n$ is the vector whose $(1,m)$ component is $\langle z_m^n \rangle \hat{\eta}_m$. $\langle \cdot \rangle$ means the average obtained using a test distribution $q(\cdot)$. The computational cost of calculating the inverse of each $\boldsymbol{W}$ is $\mathcal{O}(\mathrm{M})$ because $\boldsymbol{\Lambda}$ is defined by a tridiagonal and $\boldsymbol{L}^n$ is a diagonal matrix.

In the calculation of $q(\boldsymbol{x}^n)$, $\langle z_m^n \rangle$ controls the effective variance of the likelihood function. A higher $\langle z_m^n \rangle$ means the data are reliable for the $n$th label in the $m$th bin and lower $\langle z_m^n \rangle$ means the data are unreliable. Under the constraint $\sum_{n=1}^{N} \langle z_m^n \rangle = 1$, all labels estimate their firing rates on the basis of divide-and-conquer principle of data reliability. Using the equality $(\xi_m^n)^2 = \langle(x_m^n)^2\rangle$ that will be developed in the next section, we obtain $\hat{b}_m^n = \hat{\eta}_m \langle x_m^n \rangle - C \log 2 \cosh\langle x_m^n \rangle - \tfrac{C}{2} \log 2 \cosh(1 + (\boldsymbol{W}^n)^{-1}_{(m,m)}/\langle x_m^n \rangle^2)$ in eq. (15). When the $m$th bin includes many (few) spikes, the $n$th label tends to be selected if it estimates the highest (lowest) firing rate among the labels. But the variance of the $n$th label $(\boldsymbol{W}^n)^{-1}_{(m,m)}$ penalizes that label's selection probability.

We can also obtain the test distribution of parameters $\boldsymbol{\pi}, \boldsymbol{a}$ as

$$q(\boldsymbol{\pi}) = C(\hat{\gamma}^n) \prod_{n=1}^{N}(\pi^n)^{\hat{\gamma}^n - 1} \delta(\textstyle\sum_{n=1}^{N} \pi^n - 1), \quad (16)$$

$$q(\boldsymbol{a}) = \prod_{n=1}^{N} \left[ C(\hat{\gamma}^{nk}) \prod_{k=1}^{N}(a^{nk})^{\hat{\gamma}^{nk} - 1} \delta(\textstyle\sum_{k=1}^{N} a^{nk} - 1) \right], \quad (17)$$

where $C(\hat{\gamma}^n) = \tfrac{\Gamma(\sum_{n=1}^{N} \hat{\gamma}^n)}{\Gamma(\hat{\gamma}^1)...\Gamma(\hat{\gamma}^N)}$, $C(\hat{\gamma}^{nk}) = \tfrac{\Gamma(\sum_{k=1}^{N} \hat{\gamma}^{nk})}{\Gamma(\hat{\gamma}^{n1})...\Gamma(\hat{\gamma}^{nN})}$. $C(\hat{\gamma}^n)$ and $C(\hat{\gamma}^{nk})$ correspond to the normalization constants of $q(\boldsymbol{\pi})$ and $q(\boldsymbol{a})$, and $\hat{\gamma}^n = \langle z_1^n \rangle + \gamma^1$, $\hat{\gamma}^{nk} = \sum_{m=1}^{M-1} \langle z_m^n z_{m+1}^k \rangle + \gamma^{nk}$.

We can see $\gamma^n$ in $\hat{\gamma}^n$ controls the probability that the $n$th label is selected at the initial time, and $\gamma^{nk}$ in $\hat{\gamma}^{nk}$ biases the probability of the transition from the $n$th label to the $k$th label. A forward-backward algorithm enables us to calculate the first- and second-order statistics of $q(\boldsymbol{z})$. Since an SSSM involves many local solutions, we search for a global one using deterministic annealing, which is proven to be effective for estimating and learning in an SSSM [7].

## 3.3 EM algorithm

The EM algorithm enables us to estimate variational parameters $\boldsymbol{\xi}$ and parameters $\boldsymbol{\mu}$ and $\boldsymbol{\beta}$. In the EM algorithm, the calculation of the Q function is computationally difficult because it requires us to calculate averages using the true posterior distribution. We thus calculate the Q function using test distributions instead of the true posterior distributions as follows:

$$\tilde{Q}(\boldsymbol{\mu}, \boldsymbol{\beta}, \boldsymbol{\xi} \| \boldsymbol{\mu}^{(t')}, \boldsymbol{\beta}^{(t')}, \boldsymbol{\xi}^{(t')}) = \int d\boldsymbol{x} q(\boldsymbol{x} | \boldsymbol{\mu}^{(t')}, \boldsymbol{\beta}^{(t')}) q(\boldsymbol{z}) q(\boldsymbol{\pi}) q(\boldsymbol{a}) \log p_{\boldsymbol{\xi}}(\boldsymbol{t}, \boldsymbol{x}, \boldsymbol{z}, \boldsymbol{\pi}, \boldsymbol{a} | \boldsymbol{\mu}, \boldsymbol{\beta}). \quad (18)$$

Since $\tilde{Q}(\boldsymbol{\mu}, \boldsymbol{\beta}, \boldsymbol{\xi} \| \boldsymbol{\mu}^{(t')}, \boldsymbol{\beta}^{(t')}, \boldsymbol{\xi}^{(t')}) = -\mathcal{U}[q]_{\boldsymbol{\xi}}$, maximizing the Q function with respect to $\boldsymbol{\mu}, \boldsymbol{\beta}, \boldsymbol{\xi}$ is equivalent to minimizing the variational free energy (eq. (10) ). The update rules

$$(\xi_m^n)^2 = \langle (x_m^n)^2 \rangle, \quad \mu_m^n = \langle x_m^n \rangle, \quad \text{and} \quad \beta^n = \frac{M}{\text{Tr}[\Lambda((\mathbf{W}^n)^{-1} + (\langle \boldsymbol{x}^n \rangle - \boldsymbol{\mu}^n)(\langle \boldsymbol{x}^n \rangle - \boldsymbol{\mu}^n)^T)]} \quad (19)$$

maximize the Q function. The following table summarizes our algorithm.

```
──────────────────── Summary of our algorithm ────────────────────
Set γ¹ and γⁿᵏ.    t' ← 1    Initialize parameters of model.
Perform the following VB and EM algorithm until 𝓕_ξ[q] converges.
ξ⁽ᵗ'⁾, μ⁽ᵗ'⁾, β⁽ᵗ'⁾ ← ξ, μ, β

Variational Bayes algorithm    Perform the VB-E and VB-M step until 𝓕_ξ⁽ᵗ'⁾[q] converges.

VB-E step:    Compute q(x|μ⁽ᵗ'⁾, β⁽ᵗ'⁾) and q(z) using eq. (14) and eq. (15).
VB-M step:    Compute q(π) and q(a) using eq. (16) and eq. (17).

EM algorithm    Compute ξ, μ, β using eq. (19).

t' ← t' + 1
```

# 4 Results

The estimated firing rate in the $m$th bin is defined by $\tilde{x}_m = \langle x_m^{\tilde{n}_m} \rangle$, where $\tilde{n}_m$ satisfies $\tilde{n}_m = \arg\max_n \langle z_m^n \rangle$. The estimated change points $\tilde{m}r = \tilde{m}C\Delta$ satisfies $\langle z_{\tilde{m}}^n \rangle > \langle z_{\tilde{m}}^k \rangle$ ($\forall k \neq n$) and $\langle z_{\tilde{m}+1}^n \rangle < \langle z_{\tilde{m}+1}^k \rangle$ ($\exists k \neq n$). The estimated number of labels $\tilde{N}$ is given by $\tilde{N} = N - $ (the number of pruned labels), where we assume that the $n$th label is pruned out if $\langle z_m^n \rangle < 10^{-5} (\forall m)$. We call our algorithm "the variational Bayes switching state space model" (VB-SSSM).

## 4.1 Synthetic data analysis and Comparison with previous methods

We artificially generate spike trains from arbitrarily set firing rates with an inhomogeneous gamma process. Throughout this study, we set $\kappa$ which means the spike irregularity to 2.4 in generating spike trains. We additionally confirmed that the following results are invariant if we generate spikes using inhomogeneous Poisson or inverse Gaussian process.

In this section, we set parameters to $N = 5, T = 4000, \Delta = 0.001, r = 0.04, \gamma^n = 1, \gamma^{nk} = 100(n = k)$ or $2.5(n \neq k)$. The hyperparameters $\gamma^{nk}$ represent the *a priori* knowledge where the time scale of transitions among labels is sufficiently slower than that of firing-rate variations.

### 4.1.1 Accuracy of change-point detections

This section discusses the comparative results between the VB-SSSM and mPHMM regarding the accuracy of change-point detections and number-of-labels estimation. We used the EM algorithm to

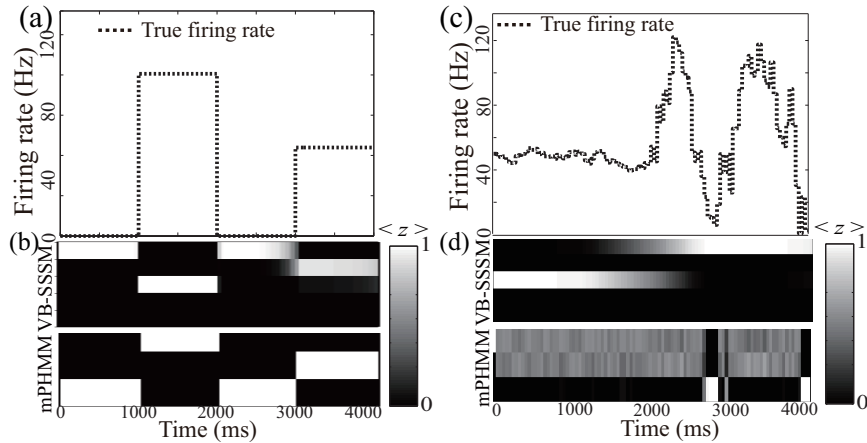

Figure 2: Comparative results of change-point detections for the VB-SSSM and the mPHMM. (a) and (c): Arbitrary set firing rates for validating the accuracy of change-point detections when firing rates include discontinuous changes in mean value (fig. (a)) or temporal correlation (fig. (c)). (b) and (d): Comparative results that correspond to firing rates in (a) ((b)) and (c) ((d)). The stronger the white color becomes, the more dominant the label is in the bin.

estimate the label variables in the mPHMM[1-3]. Since the mPHMM is useful in analyzing multi-trial data, in the estimation of mPHMM we used ten spike trains under the assumption that change points were common among ten spike trains. On the other hand, VB-SSSM uses single-trial data. Fig. 2(a) displays arbitrarily set firing rates to verify the change point detection accuracy when mean firing rates changed discontinuously. The firing rate at time $t$(ms) was set to $\lambda_t = 0.0$ $\left(t \in [0, 1000), t \in [2000, 3000)\right)$, $\lambda_t = 110.0$ $\left(t \in [1000, 2000)\right)$, and $\lambda_t = 60.0$ $\left(t \in [3000, 4000]\right)$. The upper graph in fig. 2(b) indicates the label variables estimated with the VB-SSSM and the lower indicates those estimated with the mPHMM. In the VB-SSSM, ARD estimated the number of labels to be three after pruning redundant labels. As a result of ten-trial data analysis, the VB-SSSM estimated the number of labels to be three in nine over ten spike trains. The estimated change points were 1000±0.0, 2000±0.0, and 2990±16.9ms. The true change points were 1000, 2000, and 3000ms.

Fig. 2(c) plots the arbitrarily set firing rates for verifying the change point detection accuracy when temporal correlation changes discontinuously. The firing rate at time $t$(ms) was set to $\lambda_t = \lambda_{t-1} + 2.0z_t$ $\left(t \in [0, 2000)\right)$, $\lambda_t = \lambda_{t-1} + 20.0z_t$ $\left(t \in [2000, 4000]\right)$, where $z_t$ is a standard normal random variable that satisfies $\langle z_t \rangle = 0$, $\langle z_t z_{t'} \rangle = \delta_{tt'}$ $\left(\delta_{tt'} = 0(t \neq t'), 1(t = t')\right)$. Fig. 2(d) shows the comparative results between the VB-SSSM and mPHMM. ARD estimates the number of labels to be two after pruning redundant labels. As a result of ten-trial data analysis, our algorithm estimated the number of labels to be two in nine over ten spike trains. The estimated change points was 1933±315.1ms and the true change point was 2000ms.

### 4.1.2 Accuracy of firing-rate estimation

This section discusses the nonstationary firing rate estimation accuracy. The comparative methods include kernel smoothing (KS), kernel band optimization (KBO)[17], adaptive kernel smoothing (KSA)[18], Bayesian adaptive regression splines (BARS)[19], and Bayesian binning (BB)[20]. We used a Gaussian kernel in KS, KBO, and KSA. The kernel widths $\sigma$ were set to $\sigma = 30$ (ms) (KS30), $\sigma = 50$ (ms) (KS50) and $\sigma = 100$ (ms) (KS100) in KS. In KSA, we used the bin widths estimated using KBO. Cunningham *et al.* have reviewed all of these compared methods [8].

A firing rate at time $t$(ms) was set to $\lambda_t = 5.0$ $\left(t \in [0, 480), t \in [3600, 4000]\right)$, $\lambda_t = 90.0 \times \exp(-11\frac{(t-480)}{4000})$ $\left(t \in [480, 2400)\right)$, $\lambda_t = 80.0 \times \exp(-0.5(t - 2400)/4000))$ $\left(t \in [2400, 3600)\right)$ and we reset $\lambda_t$ to 5.0 if $\lambda_t < 5.0$. We set these firing rates assuming an experiment in which transient and persistent inputs are applied to an observed neuron in a series. Note that input information, such as timings, properties, and sequences is entirely unknown.

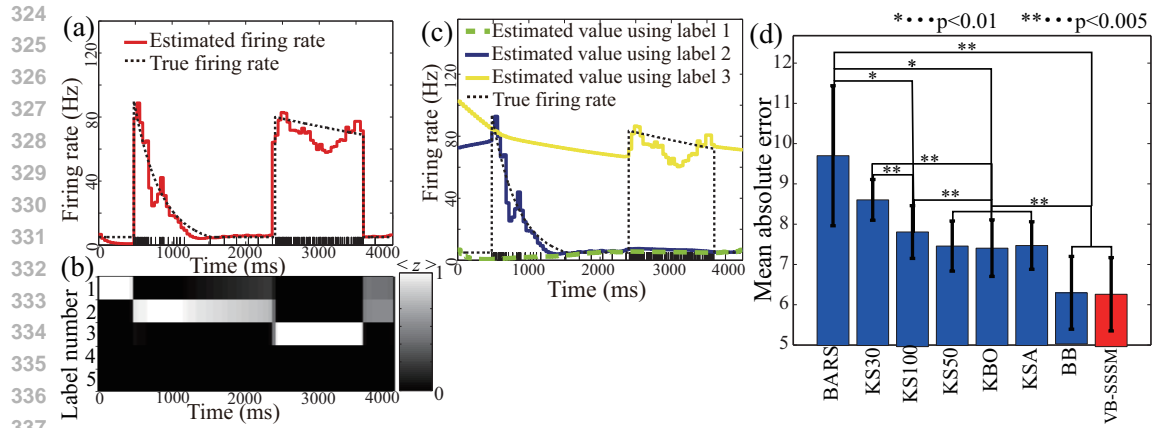

Figure 3: Results of firing-rate estimation. (a): Estimated firing rates. Vertical bars above abscissa axes are spikes used for estimates. (b): Averaged label variables $\langle z_m^n \rangle$. (c): Estimated firing rates using each label. (d): Mean absolute error $\pm$ standard deviation when applying our algorithm and other methods to estimate firing rates plotted in (a). * indicates p<0.01 and ** indicates p<0.005.

Fig. 3(a) plots the estimated firing rates (red line). Fig. 3(b) plots the estimated label variables and fig. 3(c) plots the estimated firing rates when all labels other than the pruned ones were used. ARD estimates the number of labels to be three after pruning redundant labels. As a result of ten spike trains analysis, the VB-SSSM estimated the number of labels to be three in eight over ten spike trains. The change points were estimated at $420\pm82.8$, $2385\pm20.7$, and $3605\pm14.1$ms. The true change points were 480, 2400, and 3600ms.

The mean-absolute-error (MAE) is defined by $\mathrm{MAE} = \frac{1}{K}\sum_{k=1}^{K}|\lambda_k - \hat{\lambda}_k|$, where $\lambda_k$ and $\hat{\lambda}_k$ are the true and estimated firing rates in the $k$th bin. All the methods estimate the firing rates at ten times. Fig. 3(d) shows the mean MAE values averaged across ten trials and the standard deviations. We investigated the significant differences in firing-rate estimation among all the methods using Wilcoxon signed rank test. Both the VB-SSSM and BB show the high performance. Note that the VB-SSSM can estimate not only firing rates but change points and the number of neural states.

## 4.2 Real Data Analysis

In area MT, neurons preferentially respond to the movement directions of visual inputs[21]. We analyzed the neural data recorded from area MT of a rhesus monkey when random dots were presented. These neural data are available from the Neural Signal Archive (http://www.neuralsignal.org.), and detailed experimental setups are described by Britten *et al.* [22]. The input onsets correspond to $t = 0$(ms), and the end of the recording corresponds to $t = 2000$(ms). This section discusses our analysis of the neural data included in nsa2004.1 j001 T2. These data were recorded from the same neuron of the same subject. Parameters were set as follows: $T = 2000, \Delta = 0.001, N = 5, r = 0.02, \gamma^n = 1(n = 1, ..., 5), \gamma^{nk} = 100(n = k)$ or $2.5(n \neq k)$.

Fig. 4 shows the analysis results when random dots have 3.2% coherence. Fig. 4 (a) plots the estimated firing rates (red line) and a Kolmogorov-Smirnov plot (K-S plot) (inset)[23]. Since the true firing rates for the real data are entirely unknown, we evaluated the reliability of estimated values from the confidence intervals. The black and gray lines in the inset denote the K-S plot and 95 % confidence intervals. The K-S plot supported the reliability of the estimated firing rates since it fits into the 95% confidence intervals. Fig. 4(b) depicts the estimated label variables, and fig. 4(c) shows the estimated firing rates using all labels other than the pruned ones. The VB-SSSM estimates the number of labels to be two. We call the label appearing on the right after the input onset "the 1st neural state" and that appearing after the 1st neural state "the 2nd neural state". The 1st and 2nd neural states in fig. 4 might corresponded to transient and sustained states[6] that have been heuristically detected, e.g. assuming the sustained state lasts for a constant time[24].

We analyzed all 105 spike trains recorded under presentations of random dots with 3.2%, 6.4%, 12.8%, and 99.9% coherence, precluding the neural data in which the total spike count was less than

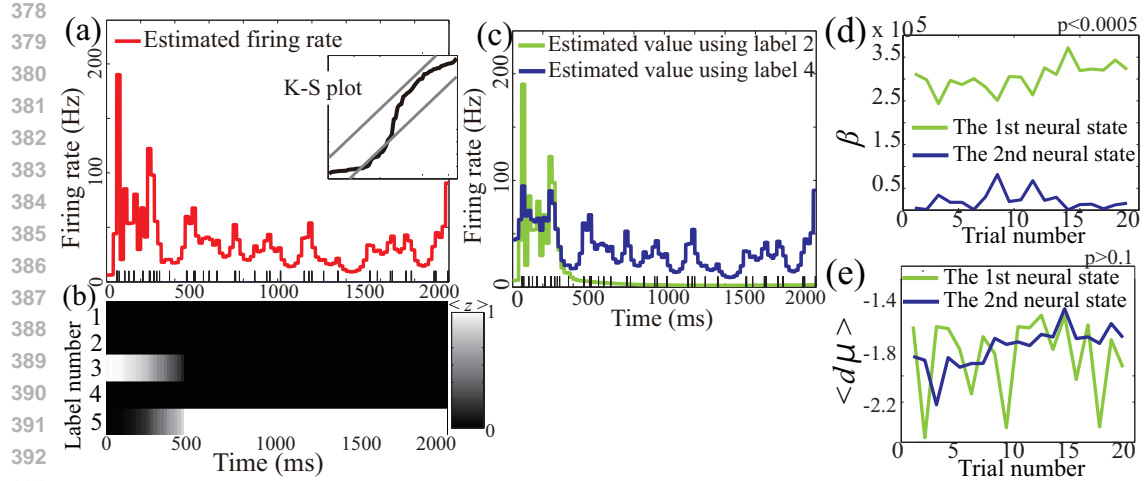

Figure 4: Estimated results when applying the VB-SSSM to area MT neural data. (a): Estimated firing rates. Vertical bars above abscissa axes are spikes used for estimates. Inset is result of Kolmogorov-Smirnov goodness-of-fit. Solid and gray lines correspond to K-S plot and 95% confidence interval. (b): Averaged label variables using test distribution. (c): Estimated firing rates using each label. (d) and (e): Estimated parameters in the 1st and the 2nd neural states.

20. The VB-SSSM estimated the number of labels to be two in 25 over 30 spike trains (3.2%), 19 over 30 spike trains (6.4%), 26 over 30 spike trains (12.8%), and 16 over 16 spike trains (99.9%). In summary, the number of labels is estimated to be two in 85 over 101 spike trains.

Figs. 4(d) and (e) show the estimated parameters from 19 spike trains whose estimated number of labels was two (6.4% coherence). The horizontal axis denotes the arranged number of trials in ascending order. Figs. 4 (d) and (e) correspond to the estimated temporal correlation $\beta$ and the time average of $\boldsymbol{\mu}$, which is defined by $\langle \boldsymbol{\mu}^n \rangle = \frac{1}{T_n} \sum_{t=1}^{T_n} \mu_t^n$ , where $T_n$ denotes the sojourn time in the $n$th label or the total observation time $T$. The estimated temporal correlation differed significantly between the 1st and 2nd neural states (Wilcoxon signed rank test, p<0.00005). On the other hand, the estimated mean firing rates did not differ significantly between these neural states (Wilcoxon signed rank test, p>0.1). Our algorithm thus detected the change points on the basis of discontinuous changes in temporal correlations. We could see the similar tendencies for all random-dot coherence conditions (data not shown). We confirmed that the mPHMM could not detect these change points (data not shown), which we were able to deduce from the results shown in fig. 2(d). These results suggest that our algorithm is effective in real data analysis.

## 5 Discussion

We proposed an algorithm for simultaneously estimating state transitions, the number of neural states, and nonstationary firing rates using single-trial data.

There are ways of extending our research to analyze multi-neuron data. The simplest one assumes that the time of state transitions is common among all recorded neurons[1-3]. Since this assumption can partially include the effect of inter-neuron interactions, we can define prior distributions that are independent between neurons. Because there are no loops in the statistical dependencies of firing rates under these conditions, the variational Bayes method can be applied directly.

One important topic for future study is optimization of coarse bin widths $r = C\Delta$. A bin width that is too wide obscures both the time of change points and temporal profile of nonstationary firing rates. A bin width that is too narrow, on the other hand, increases computational costs and worsens estimation accuracy. Watanabe *et al.* proposed an algorithm for estimating the optimal bin width by maximization the marginal likelihood [15], which is probably applicable to our algorithm.

[1] Abeles, M. et al. (1995), *PNAS*, pp. 609-616.

[2] Kemere, C. et al. (2008) *J. Neurophyiol.* **100**(7):2441-2452.

[3] Jones, L. M. et al. (2007), *PNAS* **104**(47):18772-18777.

[4] Rickert, J. et al. (2009) *J. Neurosci.* **29**(44): 13870-13882.

[5] Harvey, C. D. et al. (2009), *Nature* **461**(15):941-946.

[6] Lisberger, et al. (1999), *J. Neurosci.* **19**(6):2224-2246.

[7] Ghahramani, Z., and Hinton, G. E. (2000) *Neural Compt.* **12**(4):831-864.

[8] Cunningham J. P. et al. (2007), *Neural Netw.* **22**(9):1235-1246.

[9] Attias, H. (1999), *Proc. 15th Conf. on UAI*

[10] Beal, M. (2003), **Pd. D thesis University College London**.

[11] Jaakkola, T. S., and Jordan, M. I. (2000)., *Stat. and Compt.* **10**(1): pp. 25-37.

[12] Watanabe, K. and Okada, M. (2009) *Lecture Notes in Computer Science* **5506**:655-662.

[13] Corduneanu, A. and Bishop, C. M. (2001) *Artificial Intelligence and Statistics*: 27-34.

[14] Fuzisawa, S. et al. (2005), *Cerebral Cortex* **16**(5):639-654.

[15] Watanabe, K. et al. (2009), *IEICE* **E92-D**(7):1362-1368.

[16] Bishop, C. M. (2006), **Pattern Recognition and Machine Learning**, Springer.

[17] Shimazaki, H., and Shinomoto, S. (2007), *Neural Coding Abstract* :120-123.

[18] Richmond, B. J. et al. (1990), *J. Neurophysiol.* **64**(2):351-369.

[19] Dimatteo, I., et al. (2001), *Biometrika* **88**(4):1055-1071.

[20] Endres, D. et al. (2008), *Adv. in NIPS* **20**:393-340.

[21] Maunsell, J. H. and Van Essen, D. C. (1983) *J. Neurophysiol.* **49**(5): 1127-1147.

[22] Britten, K. H. et al. (1992), *J. Neurosci.* **12**:4745-4765.

[23] Brown, E. N. et al. (2002), *Neural Compt.* **14**(2):325-346.

[24] Bair, W. and Koch, C. (1996) *Neural Compt.* **8**(6): 1185-1202.

